# Mixture of time-warped trajectory models for movement decoding

**Elaine A. Corbett, Eric J. Perreault and Konrad P. Körding**
Northwestern University
Chicago, IL 60611
ecorbett@u.northwestern.edu

## Abstract

Applications of Brain-Machine-Interfaces typically estimate user intent based on biological signals that are under voluntary control. For example, we might want to estimate how a patient with a paralyzed arm wants to move based on residual muscle activity. To solve such problems it is necessary to integrate obtained information over time. To do so, state of the art approaches typically use a probabilistic model of how the state, e.g. position and velocity of the arm, evolves over time – a so-called trajectory model. We wanted to further develop this approach using two intuitive insights: (1) At any given point of time there may be a small set of likely movement targets, potentially identified by the location of objects in the workspace or by gaze information from the user. (2) The user may want to produce movements at varying speeds. We thus use a generative model with a trajectory model incorporating these insights. Approximate inference on that generative model is implemented using a mixture of extended Kalman filters. We find that the resulting algorithm allows us to decode arm movements dramatically better than when we use a trajectory model with linear dynamics.

## 1    Introduction

When patients have lost a limb or the ability to communicate with the outside world, brain machine interfaces (BMIs) are often used to enable robotic prostheses or restore communication. To achieve this, the user's intended *state* of the device must be decoded from biological signals. In the context of Bayesian statistics, two aspects are important for the design of an estimator of a temporally evolving state: the *observation model*, which describes how measured variables relate to the system's state and the *trajectory model* which describes how the state changes over time in a probabilistic manner. Following this logic many recent BMI applications have relied on Bayesian estimation for a wide range of problems including the decoding of intended human [1]  and animal [2] movements. In the context of BMIs, Bayesian approaches offer a principled way of formalizing the uncertainty about signals and thus often result in improvements over other signal processing techniques [1]-[3].

Most work on state estimation in dynamical systems has assumed linear dynamics and Gaussian noise. Under these circumstances, efficient algorithms result from belief propagation. The most frequent application uses the Kalman filter (KF), which recursively combines noisy state observations with the probabilistic evolution of state defined by the trajectory model to estimate the marginal distribution over states [4]. Such approaches have been used widely for applications including upper [1] and lower [5]  extremity prosthetic

devices, functional electric stimulation [6] and human computer interactions [7]. As these algorithms are so commonly used, it seems promising to develop extensions to nonlinear trajectory models that may better describe the probabilistic distribution of movements in everyday life.

One salient departure from the standard assumptions is that people tend to produce both slow and fast movements, depending on the situation. Models with linear dynamics only allow such deviation through the noise term, which makes these models poor at describing the natural variation of movement speeds during real world tasks. Explicitly incorporating movement speed into the trajectory model should lead to better movement estimates.

Knowledge of the target position should also strongly affect trajectory models. After all, we tend to accelerate our arm early during movement and slow down later on. Target information can be linearly incorporated into the trajectory model, and this has greatly improved predictions [8]-[12]. Alternatively, if there are a small number of potential targets then a mixture of trajectory models approach [13] can be used. Here we are interested in the case where available data provide a prior over potential targets but where movement targets may be anywhere. We want to incorporate target uncertainty and allow generalization to novel targets.

Prior information about potential targets could come from a number of sources but would generally be noisy. For example, activity in the dorsal premotor cortex provides information about intended target location prior to movement and may be used where such recordings are available [14]. Target information may also be found noninvasively by tracking eye movements. However, such data will generally provide non-zero priors for a number of possible target locations as the subject saccades over the scene. While subjects almost always look at a target before reaching for it [15], there may be a delay of up to a second between looking at the target and the reach – a time interval over which up to 3 saccades are typically made. Each of these fixations could be the target. Hence, a probabilistic distribution of targets is appropriate when using either neural recordings or eye tracking to estimate potential reach targets

Here we present an algorithm that uses a mixture of extended Kalman Filters (EKFs) to combine our insights related to the variation of movement speed and the availability of probabilistic target knowledge. Each of the mixture components allows the speed of the movement to vary continuously over time. We tested how well we could use EMGs and eye movements to decode hand position of humans performing a three-dimensional large workspace reaching task. We find that using a trajectory model that allows for probabilistic target information and variation of speed leads to dramatic improvements in decoding quality.

## 2      General Decoding Setting

We wanted to test how well different decoding algorithms can decode human movement, over a wide range of dynamics. While many recent studies have looked at more restrictive, two-dimensional movements, a system to restore arm function should produce a wide range of 3D trajectories. We recorded arm kinematics and EMGs of healthy subjects during unconstrained 3D reaches to targets over a large workspace. Two healthy subjects were asked to reach at slow, normal and fast speeds, as they would in everyday life. Subjects were seated as they reached towards 16 LEDs in blocks of 150s, which were located on two planes positioned such that all targets were just reachable (Fig 1A). The target LED was lit for one second prior to an auditory go cue, at which time the subject would reach to the target at the appropriate speed. Slow, normal and fast reaches were allotted 3s, 1.5s and 1s respectively; however, subjects determined the speed. An approximate total of 450 reaches were performed per subject. The subjects provided informed consent, and the protocol was approved by the Northwestern University Institutional Review Board. EMG signals were measured from the pectoralis major, and the three deltoid muscles of the shoulder. This represents a small subset of the muscles involved in reaching, and approximates those muscles retaining some voluntary control following mid-level cervical spinal cord injuries.

The EMG signals were band-pass filtered between 10 and 1,000 Hz, and subsequently anti-aliased filtered. Hand, wrist, shoulder and head positions were tracked using an Optotrak motion analysis system. We simultaneously recorded eye movements with an ASL EYETRAC-6 head mounted eye tracker.

Approximately 25% of the reaches were assigned to the test set, and the rest were used for training. Reaches for which either the motion capture data was incomplete, or there was visible motion artifact on the EMG were removed. As the state we used hand positions and joint angles (3 shoulder, 2 elbow, position, velocity and acceleration, 24 dimensions). Joint angles were calculated from the shoulder and wrist marker data using digitized bony landmarks which defined a coordinate system for the upper limb as detailed by Wu et al. [16]. As the motion data were sampled at 60Hz, the mean absolute value of the EMG in the corresponding 16.7ms windows was used as an observation of the state at each time-step. Algorithm accuracy was quantified by normalizing the root-mean-squared error by the straight line distance between the first and final position of the endpoint for each reach. We compared the algorithms statistically using repeated measures ANOVAs with Tukey post-hoc tests, treating reach and subject as random effects.

 In the rest of the paper we will ask how well these reaching movements can be decoded from EMG and eye-tracking data.

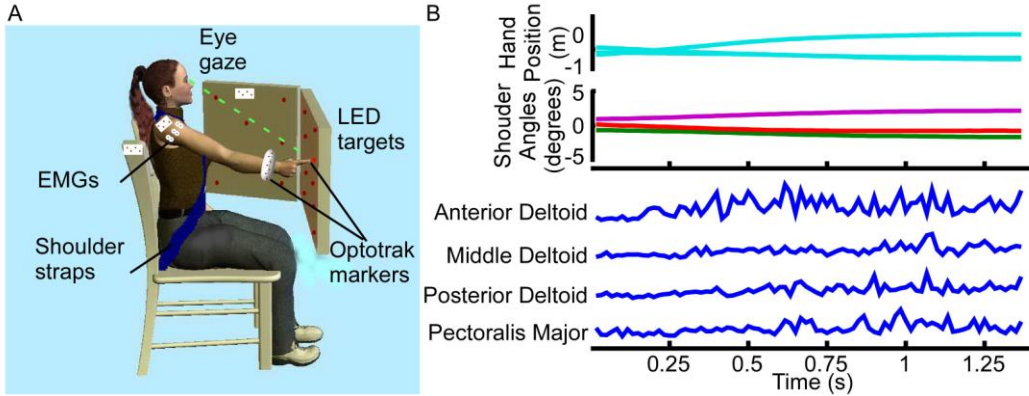

Figure 1: A Experimental setup and B sample kinematics and processed EMGs for one reach

## 3    Kalman Filters with Target information

All models that we consider in this paper assume linear observations with Gaussian noise:

$$\mathbf{y_t} = C\mathbf{x_t} + v_t, \tag{1}$$

where $\mathbf{x}$ is the state, $\mathbf{y}$ is the observation and $v$ is the measurement noise with $p(v) \sim N(0,R)$, and $R$ is the observation covariance matrix. The model fitted the measured EMGs with an average $r^2$ of 0.55. This highlights the need to integrate information over time.

The standard approach also assumes linear dynamics and Gaussian process noise:

$$\mathbf{x_t} = [x_t \, \dot{x}_t \, \ddot{x}_t]^{\mathrm{T}} = A\mathbf{x_{t-1}} + w_t, \tag{2}$$

where, $x_t \in \mathbb{R}^p$ represents the hand and joint angle positions, $w$ is the process noise with $p(w) \sim N(0,Q)$, and $Q$ is the state covariance matrix. The Kalman filter does optimal inference for this generative model.

This model can effectively capture the dynamics of stereotypical reaches to a single target by appropriately tuning its parameters. However, when used to describe reaches to multiple targets, the model cannot describe target dependent aspects of reaching but boils down to a random drift model. Fast velocities are underestimated as they are unlikely under the trajectory model and there is excessive drift close to the target (Fig. 2A).

In many decoding applications we may know the subject's target. A range of recent studies have addressed the issue of incorporating this information into the trajectory model [8, 13], and we might assume the effect of the target on the dynamics to be linear. This naturally suggests adding the target to the state space, which works well in practice [9, 12]. By appending the target to the state vector (KFT), the simple linear format of the KF may be retained:

$$\mathbf{x_t} = [x_t \; \dot{x}_t \; \ddot{x}_t \; xT_t]^\mathrm{T} = A\mathbf{x_{t-1}} + \mathbf{w_t}, \tag{3}$$

where $xT_t \in \mathbb{R}^g$ is the vector of target positions, with dimensionality less than or equal to that of $x_t$. This trajectory model thus allows describing both the rapid acceleration that characterizes the beginning of a reach and the stabilization towards its end.

We compared the accuracy of the KF and the KFT to the Single Target Model (STM), a KF trained only on reaches to the target being tested (Fig. 2). The STM represents the best possible prediction that could be obtained with a Kalman filter. Assuming the target is perfectly known, we implemented the KFT by correctly initializing the target state $xT$ at the beginning of the reach. We will relax this assumption below. The initial hand and joint angle positions were also assumed to be known.

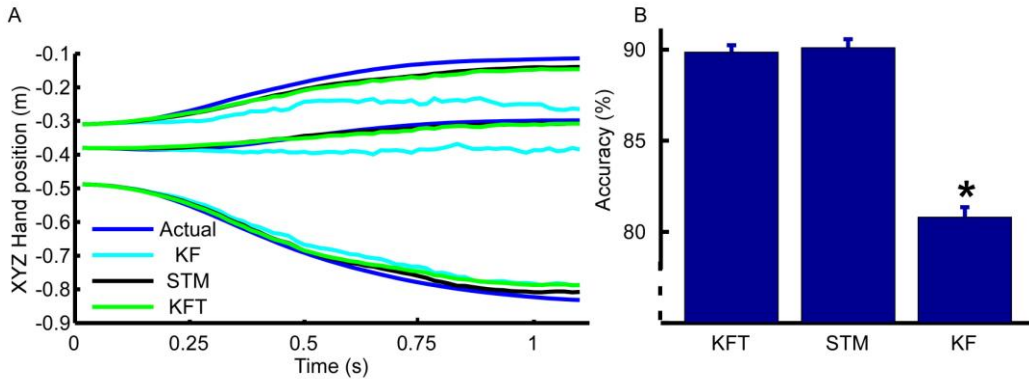

Figure 2: A Sample reach and predictions and B average accuracies with standard errors for KFT, KF and MTM.

Consistent with the recent literature, both methods that incorporated target information produced higher prediction accuracy than the standard KF (both p<0.0001). Interestingly, there was no significant difference between the KFT and the STM (p=0.9). It seems that when we have knowledge of the target, we do not lose much by training a single model over the whole workspace rather than modeling the targets individually. This is encouraging, as we desire a BMI system that can generalize to any target within the workspace, not just specifically to those that are available in the training data.

Clearly, adding the target to the state space allows the dynamics of typical movements to be modeled effectively, resulting in dramatic increases in decoding performance.

## 4    Time Warping

### 4.1    Implementing a time-warped trajectory model

While the KFT above can capture the general reach trajectory profile, it does not allow for natural variability in the speed of movements. Depending on our task objectives, which would not directly be observed by a BMI, we might lazily reach toward a target or move at maximal speed. We aim to change the trajectory model to explicitly incorporate a warping factor by which the average movement speed is scaled, allowing for such variability. As the movement speed will be positive in all practical cases, we model the logarithm of this factor,

and append it to the state vector:

$$\mathbf{x_t} = [x_t \, \dot{x}_t \, \ddot{x}_t \, xT_t \, xs_t]^{\mathrm{T}} = [x_t \, \dot{x}_t \, \ddot{x}_t \, xT_t \, \log(S_t)]^{\mathrm{T}} \tag{4}$$

We create a time-warped trajectory model by noting that if the average rate of a trajectory is to be scaled by a factor $S$, the position at time $t$ will equal that of the original trajectory at time $St$. Differentiating, the velocity will be multiplied by $S$, and the acceleration by $S^2$. For simplicity, the trajectory noise is assumed to be additive and Gaussian, and the model is assumed to be stationary:

$$\mathbf{x_t} = \begin{pmatrix} x_t \\ \dot{x}_t \\ \ddot{x}_t \\ xT_t \\ xs_t \end{pmatrix} = \begin{pmatrix} I_p & \Delta t \times I_p & 0_{p\times p} & 0_{p\times p} & 0_{p\times 1} \\ 0_{p\times p} & I_p & \Delta t \times I_p & 0_{p\times p} & 0_{p\times 1} \\ S_{t-1}^2\alpha_P & S_{t-1}\alpha_V & \alpha_A & S_{t-1}^2\alpha_T & 0_{p\times 1} \\ 0_{g\times p} & 0_{g\times p} & 0_{g\times p} & I_g & 0_{g\times 1} \\ 0_{1\times p} & 0_{1\times p} & 0_{1\times p} & 0_{1\times p} & 1 \end{pmatrix} \begin{pmatrix} x_{t-1} \\ \dot{x}_{t-1} \\ \ddot{x}_{t-1} \\ xT_{t-1} \\ xs_{t-1} \end{pmatrix} + w_t \tag{5}$$

where $I_p$ is the $p$-dimensional identity matrix and $0_{p\times p}$ is a $p\times p$ matrix of zeros. Only the $\alpha$ terms used to predict the acceleration states need to be estimated to build the state transition matrix, and they are scaled as a nonlinear function of $xs$.

After adding the variable movement speed to the state space the system is no longer linear. Therefore we need a different solution strategy. Instead of the typical KFT we use the Extended Kalman Filter (EKFT) to implement a nonlinear trajectory model by linearizing the dynamics around the best estimate at each time-step [17]. With this approach we add only small computational overhead to the KFT recursions.

## 4.2 Training the time warping model

The filter parameters were trained using a variant of the Expectation Maximization (EM) algorithm [18]. For extended Kalman filter learning the initialization for the variables may matter. $S$ was initialized with the ground truth average reach speeds for each movement relative to the average speed across all movements. The state transition parameters $\alpha$ were estimated using nonlinear least squares regression, while $C$, $Q$ and $R$ were estimated linearly for the new system, using the maximum likelihood solution [18] (M-step). For the E-step we used a standard extended Kalman smoother. We thus found the expected values for the states given the current filter parameters. For this computation, and later when testing the algorithm, $xs$ was initialized to its average value across all reaches while the remaining states were initialized to their true values. The smoothed estimate for $xs$ was then used, along with the true values for the other states, to re-estimate the filter parameters in the M-step as before. We alternated between the E and M steps until the log likelihood converged (which it did in all cases). Following the training procedure, the diagonal of the state covariance matrix Q corresponding to $xs$ was set to the variance of the smoothed $xs$ over all reaches, according to how much this state should be allowed to change during prediction. This allowed the estimate of $xs$ to develop over the course of the reach due to the evidence provided by the observations, better capturing the dynamics of reaches at different speeds.

## 4.3 Performance of the time-warped EKFT

Incorporating time warping explicitly into the trajectory model produced a noticeable increase in decoding performance over the KFT. As the speed state $xs$ is estimated throughout the course of the reach, based on the evidence provided by the observations, the trajectory model has the flexibility to follow the dynamics of the reach more accurately (Fig. 3). While at the normal self-selected speed the difference between the algorithms is small, for the slow and fast speeds, where the dynamics deviate from average, there is a clear advantage to the time warping model.

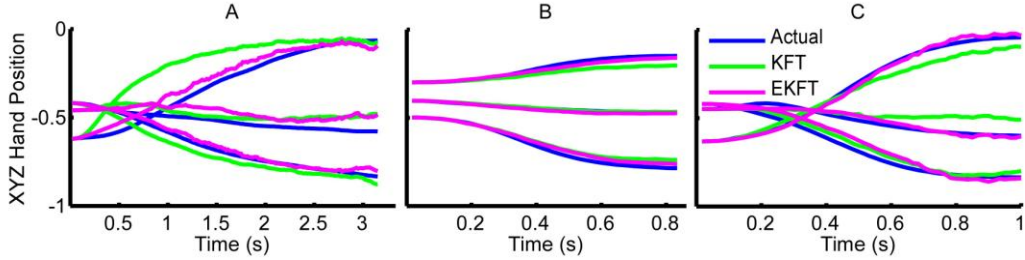

Figure 3: Hand positions and predictions of the KFT and EKFT for sample reaches at A slow, B normal and C fast speeds. Note the different time scales between reaches.

The models were first trained using data from all speeds (Fig. 4A). The EKFT was 1.8% more accurate on average (p<0.01), and the effect was significant at the slow (1.9%, p<0.05) and the fast (2.8%, p<0.01), but not at the normal (p=0.3) speed. We also trained the models from data using only reaches at the self-selected normal speed, as we wanted to see if there was enough variation to effectively train the EKFT (Fig. 4B). Interestingly, the performance of the EKFT was reduced by only 0.6%, and the KFT by 1.1%. The difference in performance between the EKFT and KFT was even more pronounced on average (2.3%, p<0.001), and for the slow and fast speeds (3.6 and 4.1%, both p< 0.0001). At the normal speed, the algorithms again were not statistically different (p=0.6). This result demonstrates that the EKFT is a practical option for a real BMI system, as it is not necessary to greatly vary the speeds while collecting training data for the model to be effective over a wide range of intended speeds.

Explicitly incorporating speed information into the trajectory model helps decoding, by modeling the natural variation in volitional speed.

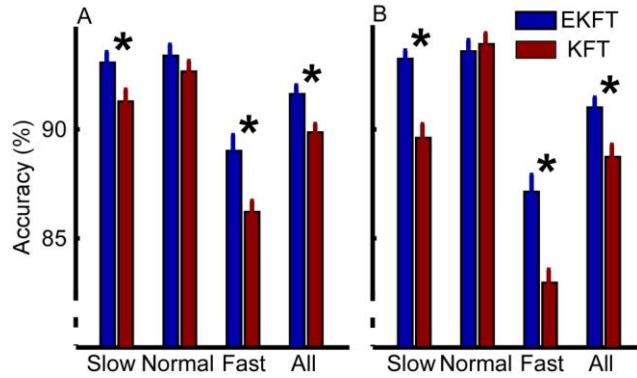

Figure 4: Mean and standard error of EKFT and KFT accuracy at the different subject-selected speeds. Models were trained on reaches at A all speeds and B just normal speed reaches. Asterisks indicate statistically significant differences between the algorithms.

## 5    Mixtures of Targets

So far, we have assumed that the targets of our reaches are perfectly known. In a real-world system, there will be uncertainty about the intended target of the reach. However, in typical applications there are a small number of possible objectives. Here we address this situation. Drawing on the recent literature, we use a mixture model to consider each of the possible targets [11, 13]. We condition the posterior probability for the state on the N possible targets, $T$:

$$P(\mathbf{x_t}|\mathbf{y_{1...t}}) = \sum_{T=T1}^{TN} P(\mathbf{x_t}|\mathbf{y_{1...t}}, \mathbf{xT_t}) P(\mathbf{xT_t}|\mathbf{y_{1...t}}) \tag{6}$$

Using Bayes' Rule, this equation becomes:

$$P(\mathbf{x_t}|\mathbf{y_{1\ldots t}}) = \sum_{T=T1}^{TK} P(\mathbf{x_t}|\mathbf{y_{1\ldots t}}, \boldsymbol{xT_t}) \frac{P(\boldsymbol{y_{1\ldots t}}|\boldsymbol{xT_t})P(\boldsymbol{xT_t})}{P(\boldsymbol{y_{1\ldots t}})} \qquad (7)$$

As we are dealing with a mixture model, we perform the Kalman filter recursion for each possible target, $xT$, and our solution is a weighted sum of the outputs. The weights are proportional to the prior for that target, $P(\boldsymbol{xT_t})$, and the likelihood of the model given that target $P(\boldsymbol{y_{1\ldots t}}|\boldsymbol{xT_t})$. $P(\boldsymbol{y_{1\ldots t}})$ is independent of the target and does not need to be calculated.

We tested mixtures of both algorithms, the mKFT and mEKFT, with real uncertain priors obtained from eye-tracking in the one-second period preceding movement. As the targets were situated on two planes, the three-dimensional location of the eye gaze was found by projecting its direction onto those planes. The first, middle and last eye samples were selected, and all other samples were assigned to a group according to which of the three was closest. The mean and variance of these three groups were used to initialize three Kalman filters in the mixture model. The priors of the three groups were assigned proportional to the number of samples in them. If the subject looks at multiple positions prior to reaching, this method ensures with a high probability that the correct target was accounted for in one of the filters in the mixture.

We also compared the MTM approach of Yu et al. [13], where a different KF model was generated for each target, and a mixture is performed over these models. This approach explicitly captures the dynamics of stereotypical reaches to specific targets. Given perfect target information, it would reduce to the STM described above. Priors for the MTM were found by assigning each valid eye sample to its closest two targets, and weighting the models proportional to the number of samples assigned to the corresponding target, divided by its distance from the mean of those samples. We tried other ways of assigning priors and the one presented gave the best results.

We calculated the reduction in decoding quality when instead of perfect priors we provide eye-movement based noisy priors (Fig. 5). The accuracies of the mEKFT, the mKFT and the MTM were only degraded by 0.8, 1.9 and 2.1% respectively, compared to the perfect prior situation. The mEKFT was still close to 10% better than the KF. The mixture model framework is effective in accounting for uncertain priors.

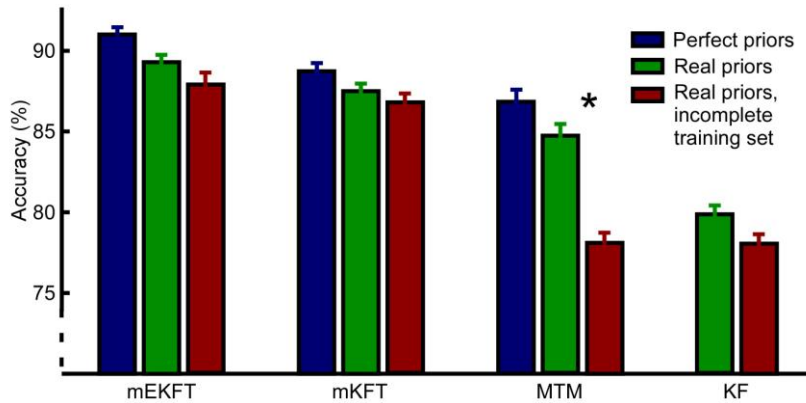

Figure 5: Mean and standard errors of accuracy for algorithms with perfect priors, and uncertain priors with full and partial training set. The asterisk indicates a statistically significant effects between the two training types, where real priors are used.

Here, only reaches at normal speed were used to train the models, as this is a more realistic training set for a BMI application. This accounts for the degraded performance of the MTM with perfect priors relative to the STM from above (Fig. 2). With even more stereotyped training data for each target, the MTM doesn't generalize as well to new speeds.

We also wanted to know if the algorithms could generalize to new targets. In a real application, the available training data will generally not span the entire useable workspace. We compared the algorithms where reaches to all targets except the one being tested had been used to train the models. The performance of the MTM was significantly degraded unsurprisingly, as it was designed for reaches to a set of known targets. Performance of the mKFT and mEKFT degraded by about 1%, but not significantly (both $p>0.7$), demonstrating that the continuous approach to target information is preferable when the target could be anywhere in space, not just at locations for which training data is available.

# 6    Discussion and conclusions

The goal of this work was to design a trajectory model that would improve decoding for BMIs with an application to reaching. We incorporated two features that prominently influence the dynamics of natural reach: the movement speed and the target location. Our approach is appropriate where uncertain target information is available. The model generalizes well to new regions of the workspace for which there is no training data, and across a broad range of reaching dynamics to widely spaced targets in three dimensions.

The advantages over linear models in decoding precision we report here could be equally obtained using mixtures over many targets and speeds. While mixture models [11, 13] could allow for slow versus fast movements and any number of potential targets, this strategy will generally require many mixture components. Such an approach would require a lot more training data, as we have shown that it does not generalize well. It would also be run-time intensive which is problematic for prosthetic devices that rely on low power controllers. In contrast, the algorithm introduced here only takes a small amount of additional run-time in comparison to the standard KF approach. The EKF is only marginally slower than the standard KF and the algorithm will not generally need to consider more than 3 mixture components assuming the subject fixates the target within the second preceding the reach.

In this paper we assumed that subjects always would fixate a reach target – along with other non-targets. While this is close to the way humans usually coordinate eyes and reaches [15], there might be cases where people do not fixate a reach target. Our approach could be easily extended to deal with such situations by adding a dummy mixture component that allows the description of movements to any target.

As an alternative to mixture approaches, a system can explicitly estimate the target position in the state vector [9]. This approach, however, would not straightforwardly allow for the rich target information available; we look at the target but also at other locations, strongly suggesting mixture distributions. A combination of the two approaches could further improve decoding quality. We could both estimate speed and target position for the EKFT in a continuous manner while retaining the mixture over target priors.

We believe that the issues that we have addressed here are almost universal. Virtually all types of movements are executed at varying speed. A probabilistic distribution for a small number of action candidates may also be expected in most BMI applications – after all there are usually only a small number of actions that make sense in a given environment. While this work is presented in the context of decoding human reaching, it may be applied to a wide range of BMI applications including lower limb prosthetic devices and human computer interactions, as well as different signal sources such as electrode grid recordings and electroencephalograms. The increased user control in conveying their intended movements would significantly improve the functionality of a neuroprosthetic device.

**Acknowledgements**

The authors thank T. Haswell, E. Krepkovich, and V. Ravichandran for assistance with experiments. This work was funded by the NSF Program in Cyber-Physical Systems.

**References**

[1]    L.R. Hochberg, M.D. Serruya, G.M. Friehs, J.A. Mukand, M. Saleh, A.H. Caplan, A. Branner, D.

Chen, R.D. Penn, and J.P. Donoghue, "Neuronal ensemble control of prosthetic devices by a human with tetraplegia," *Nature*, vol. 442, 2006, pp. 164–171.

[2]   W. Wu, Y. Gao, E. Bienenstock, J.P. Donoghue, and M.J. Black, "Bayesian population decoding of motor cortical activity using a Kalman filter," *Neural Computation*, vol. 18, 2006, pp. 80–118.

[3]   W. Wu, M.J. Black, Y. Gao, E. Bienenstock, M. Serruya, A. Shaikhouni, and J.P. Donoghue, "Neural decoding of cursor motion using a Kalman filter," *Advances in Neural Information Processing Systems 15: Proceedings of the 2002 Conference*, 2003, p. 133.

[4]   R.E. Kalman, "A new approach to linear filtering and prediction problems," *Journal of basic Engineering*, vol. 82, 1960, pp. 35–45.

[5]   G.G. Scandaroli, G.A. Borges, J.Y. Ishihara, M.H. Terra, A.F.D. Rocha, and F.A.D.O. Nascimento, "Estimation of foot orientation with respect to ground for an above knee robotic prosthesis," *Proceedings of the 2009 IEEE/RSJ international conference on Intelligent robots and systems*, St. Louis, MO, USA: IEEE Press, 2009, pp. 1112-1117.

[6]   I. Cikajlo, Z. Matjačić, and T. Bajd, "Efficient FES triggering applying Kalman filter during sensory supported treadmill walking," *Journal of Medical Engineering & Technology*, vol. 32, 2008, pp. 133-144.

[7]   S. Kim, J.D. Simeral, L.R. Hochberg, J.P. Donoghue, and M.J. Black, "Neural control of computer cursor velocity by decoding motor cortical spiking activity in humans with tetraplegia," *Journal of Neural Engineering*, vol. 5, 2008, pp. 455-476.

[8]   L. Srinivasan, U.T. Eden, A.S. Willsky, and E.N. Brown, "A state-space analysis for reconstruction of goal-directed movements using neural signals," *Neural computation*, vol. 18, 2006, pp. 2465–2494.

[9]   G.H. Mulliken, S. Musallam, and R.A. Andersen, "Decoding trajectories from posterior parietal cortex ensembles," *Journal of Neuroscience*, vol. 28, 2008, p. 12913.

[10]  W. Wu, J.E. Kulkarni, N.G. Hatsopoulos, and L. Paninski, "Neural Decoding of Hand Motion Using a Linear State-Space Model With Hidden States," *IEEE Transactions on neural systems and rehabilitation engineering*, vol. 17, 2009, p. 1.

[11]  J.E. Kulkarni and L. Paninski, "State-space decoding of goal-directed movements," *IEEE Signal Processing Magazine*, vol. 25, 2008, p. 78.

[12]  C. Kemere and T. Meng, "Optimal estimation of feed-forward-controlled linear systems," *IEEE International Conference on Acoustics, Speech, and Signal Processing, 2005. Proceedings.(ICASSP'05)*, 2005.

[13]  B.M. Yu, C. Kemere, G. Santhanam, A. Afshar, S.I. Ryu, T.H. Meng, M. Sahani, and K.V. Shenoy, "Mixture of trajectory models for neural decoding of goal-directed movements," *Journal of neurophysiology*, vol. 97, 2007, p. 3763.

[14]  N. Hatsopoulos, J. Joshi, and J.G. O'Leary, "Decoding continuous and discrete motor behaviors using motor and premotor cortical ensembles," *Journal of neurophysiology*, vol. 92, 2004, p. 1165.

[15]  R.S. Johansson, G. Westling, A. Backstrom, and J.R. Flanagan, "Eye-hand coordination in object manipulation," *Journal of Neuroscience*, vol. 21, 2001, p. 6917.

[16]  G. Wu, F.C. van der Helm, H.E.J. Veeger, M. Makhsous, P. Van Roy, C. Anglin, J. Nagels, A.R. Karduna, and K. McQuade, "ISB recommendation on definitions of joint coordinate systems of various joints for the reporting of human joint motion–Part II: shoulder, elbow, wrist and hand," *Journal of biomechanics*, vol. 38, 2005, pp. 981–992.

[17]  D. Simon, *Optimal state estimation: Kalman, H [infinity] and nonlinear approaches*, John Wiley and Sons, 2006.

[18]  Z. Ghahramani and G.E. Hinton, "Parameter estimation for linear dynamical systems," *University of Toronto technical report CRG-TR-96-2*, vol. 6, 1996.

